# Predictive Representations of State

**Michael L. Littman**
**Richard S. Sutton**
AT&T Labs–Research, Florham Park, New Jersey
{mlittman,sutton}@research.att.com

**Satinder Singh**
Syntek Capital, New York, New York
baveja@cs.colorado.edu

## Abstract

We show that states of a dynamical system can be usefully represented by multi-step, action-conditional predictions of future observations. State representations that are grounded in data in this way may be easier to learn, generalize better, and be less dependent on accurate prior models than, for example, POMDP state representations. Building on prior work by Jaeger and by Rivest and Schapire, in this paper we compare and contrast a linear specialization of the predictive approach with the state representations used in POMDPs and in $k$-order Markov models. Ours is the first specific formulation of the predictive idea that includes both stochasticity and actions (controls). We show that any system has a linear predictive state representation with number of predictions no greater than the number of states in its minimal POMDP model.

In predicting or controlling a sequence of observations, the concepts of state and state estimation inevitably arise. There have been two dominant approaches. The *generative-model* approach, typified by research on partially observable Markov decision processes (POMDPs), hypothesizes a structure for generating observations and estimates its state and state dynamics. The *history-based* approach, typified by $k$-order Markov methods, uses simple functions of past observations as state, that is, as the immediate basis for prediction and control. (The data flow in these two approaches are diagrammed in Figure 1.) Of the two, the generative-model approach is more general. The model's internal state gives it temporally unlimited memory—the ability to remember an event that happened arbitrarily long ago—whereas a history-based approach can only remember as far back as its history extends. The bane of generative-model approaches is that they are often strongly dependent on a good model of the system's dynamics. Most uses of POMDPs, for example, assume a perfect dynamics model and attempt only to estimate state. There are algorithms for simultaneously estimating state and dynamics (e.g., Chrisman, 1992), analogous to the Baum-Welch algorithm for the uncontrolled case (Baum et al., 1970), but these are only effective at tuning parameters that are already approximately correct (e.g., Shatkay & Kaelbling, 1997).

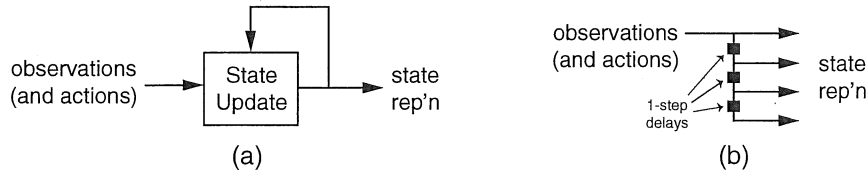

Figure 1: Data flow in a) POMDP and other recursive updating of state representation, and b) history-based state representation.

In practice, history-based approaches are often much more effective. Here, the state representation is a relatively simple record of the stream of past actions and observations. It might record the occurrence of a specific subsequence or that one event has occurred more recently than another. Such representations are far more closely linked to the data than are POMDP representations. One way of saying this is that POMDP learning algorithms encounter many local minima and saddle points because all their states are equipotential. History-based systems immediately break symmetry, and their direct learning procedure makes them comparably simple. McCallum (1995) has shown in a number of examples that sophisticated history-based methods can be effective in large problems, and are often more practical than POMDP methods even in small ones.

The *predictive state representation* (PSR) approach, which we develop in this paper, is like the generative-model approach in that it updates the state representation recursively, as in Figure 1(a), rather than directly computing it from data. We show that this enables it to attain generality and compactness at least equal to that of the generative-model approach. However, the PSR approach is also like the history-based approach in that its representations are grounded in data. Whereas a history-based representation looks to the past and records what did happen, a PSR looks to the future and represents what *will* happen. In particular, a PSR is a vector of predictions for a specially selected set of action–observation sequences, called *tests* (after Rivest & Schapire, 1994). For example, consider the test $a_1o_1a_2o_2$, where $a_1$ and $a_2$ are specific actions and $o_1$ and $o_2$ are specific observations. The correct prediction for this test given the data stream up to time $k$ is the probability of its observations occurring (in order) given that its actions are taken (in order) (i.e., $Pr\{O_k = o_1, O_{k+1} = o_2 \mid A_k = a_1, A_{k+1} = a_2\}$). Each test is a kind of experiment that could be performed to tell us something about the system. If we knew the outcome of all possible tests, then we would know everything there is to know about the system. A PSR is a set of tests that is sufficient information to determine the prediction for all possible tests (a sufficient statistic).

As an example of these points, consider the float/reset problem (Figure 2) consisting of a linear string of 5 states with a distinguished *reset* state on the far right. One action, f (float), causes the system to move uniformly at random to the right or left by one state, bounded at the two ends. The other action, r (reset), causes a jump to the reset state irrespective of the current state. The observation is always 0 unless the r action is taken when the system is already in the reset state, in which case the observation is 1. Thus, on an f action, the correct prediction is always 0, whereas on an r action, the correct prediction depends on how many fs there have been since the last r: for zero fs, it is 1; for one or two fs, it is 0.5; for three or four fs, it is 0.375; for five or six fs, it is 0.3125, and so on decreasing after every second f, asymptotically bottoming out at 0.2.

No $k$-order Markov method can model this system exactly, because no limited-

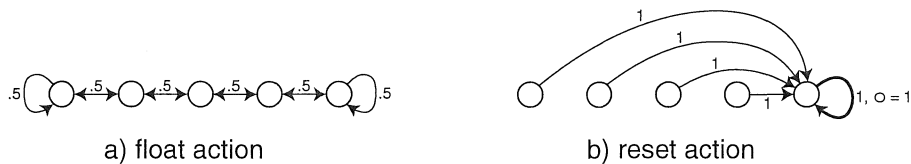

| a) float action | b) reset action |

Figure 2: Underlying dynamics of the float/reset problem for a) the float action and b) the reset action. The numbers on the arcs indicate transition probabilities. The observation is always 0 except on the reset action from the rightmost state, which produces an observation of 1.

length history is a sufficient statistic. A POMDP approach can model it exactly by maintaining a belief-state representation over five or so states. A PSR, on the other hand, can exactly model the float/reset system using just two tests: r1 and f0r1. Starting from the rightmost state, the correct predictions for these two tests are always two successive probabilities in the sequence given above (1, 0.5, 0.5, 0.375,...), which is always a sufficient statistic to predict the next pair in the sequence. Although this informational analysis indicates a solution is possible in principle, it would require a nonlinear updating process for the PSR.

In this paper we restrict consideration to a linear special case of PSRs, for which we can guarantee that the number of tests needed does not exceed the number of states in the minimal POMDP representation (although we have not ruled out the possibility it can be considerably smaller). Of greater ultimate interest are the prospects for learning PSRs and their update functions, about which we can only speculate at this time. The difficulty of learning POMDP structures without good prior models are well known. To the extent that this difficulty is due to the indirect link between the POMDP states and the data, predictive representations may be able to do better.

Jaeger (2000) introduced the idea of predictive representations as an alternative to belief states in hidden Markov models and provided a learning procedure for these models. We build on his work by treating the control case (with actions), which he did not significantly analyze. We have also been strongly influenced by the work of Rivest and Schapire (1994), who did consider tests including actions, but treated only the deterministic case, which is significantly different. They also explored construction and learning algorithms for discovering system structure.

# 1 Predictive State Representations

We consider dynamical systems that accept actions from a discrete set $\mathcal{A}$ and generate observations from a discrete set $\mathcal{O}$. We consider only predicting the system, not controlling it, so we do not designate an explicit reward observation. We refer to such a system as an *environment*. We use the term *history* to denote a test forming an initial stream of experience and characterize an environment by a probability distribution over all possible histories, $P : \{\mathcal{O}|\mathcal{A}\}^* \mapsto [0, 1]$, where $P(o_1 \cdots o_\ell | a_1 \cdots a_\ell)$ is the probability of observations $o_1, \ldots, o_\ell$ being generated, in that order, given that actions $a_1, \ldots, a_\ell$ are taken, in that order. The probability of a test $t$ conditional on a history $h$ is defined as $P(t|h) = P(ht)/P(h)$. Given a set of $q$ tests $Q = \{t_i\}$, we define their $(1 \times q)$ *prediction vector*, $p(h) = [P(t_1|h), P(t_2|h), \ldots, P(t_q|h)]$, as a predictive state representation (PSR) if and only if it forms a sufficient statistic for the environment, i.e., if and only if

$$P(t|h) = f_t(p(h)), \tag{1}$$

for any test $t$ and history $h$, and for some *projection function* $f_t : [0,1]^q \mapsto [0,1]$. In this paper we focus on *linear PSRs*, for which the projection functions are linear, that is, for which there exist a $(1 \times q)$ *projection vector* $m_t$, for every test $t$, such that

$$P(t|h) = f_t(p(h)) = p(h)m_t^T, \qquad (2)$$

for all histories $h$.

Let $p_i(h)$ denote the $i$th component of the prediction vector for some PSR. This can be updated recursively, given a new action–observation pair $a, o$, by

$$p_i(hao) = P(t_i|hao) = \frac{P(ot_i|ha)}{P(o|ha)} = \frac{f_{aot_i}(p(h))}{f_{ao}(p(h))} = \frac{p(h)m_{aot_i}^T}{p(h)m_{ao}^T}, \qquad (3)$$

where the last step is specific to linear PSRs. We can now state our main result:

**Theorem 1** *For any environment that can be represented by a finite POMDP model, there exists a linear PSR with number of tests no larger than the number of states in the minimal POMDP model.*

## 2 Proof of Theorem 1: Constructing a PSR from a POMDP

We prove Theorem 1 by showing that for any POMDP model of the environment, we can construct in polynomial time a linear PSR for that POMDP of lesser or equal complexity that produces the same probability distribution over histories as the POMDP model.

We proceed in three steps. First, we review POMDP models and how they assign probabilities to tests. Next, we define an algorithm that takes an $n$-state POMDP model and produces a set of $n$ or fewer tests, each of length less than or equal to $n$. Finally, we show that the set of tests constitute a PSR for the POMDP, that is, that there are projection vectors that, together with the tests' predictions, produce the same probability distribution over histories as the POMDP.

A POMDP (Lovejoy, 1991; Kaelbling et al., 1998) is defined by a sextuple $\langle S, \mathcal{A}, \mathcal{O}, b_0, T, O \rangle$. Here, $S$ is a set of $n$ underlying (hidden) states, $\mathcal{A}$ is a discrete set of actions, and $\mathcal{O}$ is a discrete set of observations. The $(1 \times n)$ vector $b_0$ is an initial state distribution. The set $T$ consists of $(n \times n)$ transition matrices $T^a$, one for each action $a$, where $T_{ij}^a$ is the probability of a transition from state $i$ to $j$ when action $a$ is chosen. The set $O$ consists of diagonal $(n \times n)$ observation matrices $O^{a,o}$, one for each pair of observation $o$ and action $a$, where $O_{ii}^{a,o}$ is the probability of observation $o$ when action $a$ is selected and state $i$ is reached.[1]

The state representation in a POMDP (Figure 1(a)) is the *belief state*—the $(1 \times n)$ vector of the state-occupation probabilities given the history $h$. It can be computed recursively given a new action $a$ and observation $o$ by

$$b(hao) = \frac{b(h)T^a O^{a,o}}{b(h)T^a O^{a,o} e_n^T},$$

where $e_n$ is the $(1 \times n)$-vector of all 1s.

Finally, a POMDP defines a probability distribution over tests (and thus histories) by

$$P(o_1 \cdots o_\ell | h a_1 \cdots a_\ell) = b(h)T^{a_1}O^{a_1,o_1} \cdots T^{a_\ell}O^{a_\ell,o_\ell}e_n^T. \qquad (4)$$

We now present our algorithm for constructing a PSR for a given POMDP. It uses a function $u$ mapping tests to $(1 \times n)$ vectors defined recursively by $u(\varepsilon) = e_n$ and $u(aot) = (T^a O^{a,o} u(t)^T)^T$, where $\varepsilon$ represents the null test. Conceptually, the components of $u(t)$ are the probabilities of the test $t$ when applied from each underlying state of the POMDP; we call $u(t)$ the *outcome vector* for test $t$. We say a test $t$ is *linearly independent* of a set of tests $S$ if its outcome vector is linearly independent of the set of outcome vectors of the tests in $S$. Our algorithm `search` is used and defined as

$$Q \leftarrow \texttt{search}(\varepsilon, \{\})$$

```
search(t, S):
    for each a ∈ A, o ∈ O
        if aot is linearly independent of S
            then S ← search(aot, S ∪ {aot})
    return S
```

The algorithm maintains a set of tests and searches for new tests that are linearly independent of those already found. It is a form of depth-first search. The algorithm halts when it checks all the one-step extensions of its tests and finds none that are linearly independent. Because the set of tests $Q$ returned by `search` have linearly independent outcome vectors, the cardinality of $Q$ is bounded by $n$, ensuring that the algorithm halts after a polynomial number of iterations. Because each test in $Q$ is formed by a one-step extension to some other test in $Q$, no test is longer than $n$ action–observation pairs.

The check for linear independence can be performed in many ways, including Gaussian elimination, implying that `search` terminates in polynomial time.

By construction, all one-step extensions to the set of tests $Q$ returned by `search` are linearly dependent on those in $Q$. We now show that this is true for any test.

**Lemma 1** *The outcome vectors of the tests in $Q$ can be linearly combined to produce the outcome vector for any test.*

**Proof:** Let $U$ be the $(n \times q)$ matrix formed by concatenating the outcome vectors for all tests in $Q$. Since, for all combinations of $a$ and $o$, the columns of $T^a O^{a,o} U$ are linearly dependent on the columns of $U$, we can write $T^a O^{a,o} U = U W^T$ for some $q \times q$ matrix of weights $W$.

If $t$ is a test that is linearly dependent on $Q$, then any one-step extension of $t$, $aot$, is linearly dependent on $Q$. This is because we can write the outcome vector for $t$ as $u(t) = (U w^T)^T$ for some $(1 \times q)$ weight vector $w$ and the outcome vector for $aot$ as $u(aot) = (T^a O^{a,o} u(t)^T)^T = (T^a O^{a,o} U w^T)^T = (U W^T w^T)^T$. Thus, $aot$ is linearly dependent on $Q$.

Now, note that all one-step tests are linearly dependent on $Q$ by the structure of the `search` algorithm. Using the previous paragraph as an inductive argument, this implies that *all* tests are linearly dependent on $Q$. □

Returning to the float/reset example POMDP, `search` begins with by enumerating the 4 extensions to the null test (`f0`, `f1`, `r0`, and `r1`). Of these, only `f0` and `r0` are are linearly independent. Of the extensions of these, `f0r0` is the only one that is linearly independent of the other two. The remaining two tests added to $Q$ by `search` are `f0f0r0` and `f0f0f0r0`. No extensions of the 5 tests in $Q$ are linearly independent of the 5 tests in $Q$, so the procedure halts.

We now show that the set of tests $Q$ constitute a PSR for the POMDP by constructing projection vectors that, together with the tests' predictions, produce the same probability distribution over histories as the POMDP.

For each combination of $a$ and $o$, define a $q \times q$ matrix $M_{ao} = (U^+ T^a O^{a,o} U)^T$ and a $1 \times q$ vector $m_{ao} = (U^+ T^a O^{a,o} e_n^T)^T$, where $U$ is the matrix of outcome vectors defined in the previous section and $U^+$ is its pseudoinverse[2]. The $i$th row of $M_{ao}$ is $m_{aot_i}$. The probability distribution on histories implied by these projection vectors is

$$
\begin{aligned}
P(o_1 \cdots o_\ell | h a_1 \cdots a_\ell) &= p(h) m_{a_1 o_1 \cdots a_\ell o_\ell}^T \\
&= p(h) M_{a_1 o_1}^T \cdots M_{a_{\ell-1} o_{\ell-1}}^T m_{a_\ell o_\ell}^T \\
&= b(h) U U^+ T^{a_1} O^{a_1, o_1} U \cdots U^+ T^{a_{\ell-1}} O^{a_{\ell-1}, o_{\ell-1}} U U^+ T^{a_\ell} O^{a_\ell, o_\ell} e_n^T \\
&= b(h) T^{a_1} O^{a_1, o_1} \cdots T^{a_{\ell-1}} O^{a_{\ell-1}, o_{\ell-1}} T^{a_\ell} O^{a_\ell, o_\ell} e_n^T,
\end{aligned}
$$

i.e., it is the same as that of the POMDP, as in Equation 4. Here, the last step uses the fact that $U U^+ v^T = v^T$ for $v^T$ linearly dependent on the columns of $U$. This holds by construction of $U$ in the previous section.

This completes the proof of Theorem 1.

Completing the float/reset example, consider the $M_{\mathbf{f},0}$ matrix found by the process defined in this section. It derives predictions for each test in $Q$ after taking action $\mathbf{f}$. Most of these are quite simple because the tests are so similar: the new prediction for $\mathbf{r0}$ is exactly the old prediction for $\mathbf{f0r0}$, for example. The only non trivial test is $\mathbf{f0f0f0r0}$. Its outcome can be computed from $0.250\, p(\mathbf{r0}|h) - 0.0625\, p(\mathbf{f0r0}|h) + 0.750\, p(\mathbf{f0f0r0}|h)$. This example illustrates that the projection vectors need not contain only positive entries.

## 3    Conclusion

We have introduced a predictive state representation for dynamical systems that is grounded in actions and observations and shown that, even in its linear form, it is at least as general and compact as POMDPs. In essence, we have established PSRs as a non-inferior alternative to POMDPs, and suggested that they might have important advantages, while leaving demonstration of those advantages to future work. We conclude by summarizing the potential advantages (to be explored in future work):

**Learnability.** The $k$-order Markov model is similar to PSRs in that it is entirely based on actions and observations. Such models can be learned trivially from data by counting—it is an open question whether something similar can be done with a PSR. Jaeger (2000) showed how to learn such a model in the uncontrolled setting, but the situation is more complex in the multiple action case since outcomes are conditioned on behavior, violating some required independence assumptions.

**Compactness.** We have shown that there exist linear PSRs no more complex that the minimal POMDP for an environment, but in some cases the minimal linear PSR seems to be much smaller. For example, a POMDP extension of *factored MDPs* explored by Singh and Cohn (1998) would be cross-products of separate POMDPs and have linear PSRs that increase linearly with the number and size of the component POMDPs, whereas their minimal POMDP representation would grow as the size

of the state space, i.e., exponential in the number of component POMDPs. This (apparent) advantage stems from the PSR's combinatorial or factored structure. As a vector of state *variables*, capable of taking on diverse values, a PSR may be inherently more powerful than the distribution over discrete states (the belief state) of a POMDP. We have already seen that general PSRs can be more compact than POMDPs; they are also capable of efficiently capturing environments in the diversity representation used by Rivest and Schapire (1994), which is known to provide an extremely compact representation for some environments.

**Generalization.** There are reasons to think that state variables that are themselves predictions may be particularly useful in learning to make other predictions. With so many things to predict, we have in effect a set or sequence of learning problems, all due to the same environment. In many such cases the solutions to earlier problems have been shown to provide features that generalize particularly well to subsequent problems (e.g., Baxter, 2000; Thrun & Pratt, 1998).

**Powerful, extensible representations.** PSRs that predict tests could be generalized to predict the outcomes of multi-step options (e.g., Sutton et al., 1999). In this case, particularly, they would constitute a powerful language for representing the state of complex environments.

**Acknowledgments:** We thank Peter Dayan, Lawrence Saul, Fernando Pereira and Rob Schapire for many helpful discussions of these and related ideas.

## Footnotes

[1]There are many equivalent formulations and the conversion procedure described here can be easily modified to accommodate other POMDP definitions.

[2]If $U = A \Sigma B^T$ is the singular value decomposition of $U$, then $B \Sigma^+ A^T$ is the pseudoinverse. The pseudoinverse of the diagonal matrix $\Sigma$ replaces each non-zero element with its reciprocal.

# References

Baum, L. E., Petrie, T., Soules, G., & Weiss, N. (1970). A maximization technique occurring in the statistical analysis of probabilistic functions of Markov chains. *Annals of Mathematical Statistics, 41*, 164–171.

Baxter, J. (2000). A model of inductive bias learning. *Journal of Artificial Intelligence Research, 12*, 149–198.

Chrisman, L. (1992). Reinforcement learning with perceptual aliasing: The perceptual distinctions approach. *Proceedings of the Tenth National Conference on Artificial Intelligence* (pp. 183–188). San Jose, California: AAAI Press.

Jaeger, H. (2000). Observable operator models for discrete stochastic time series. *Neural Computation, 12*, 1371–1398.

Kaelbling, L. P., Littman, M. L., & Cassandra, A. R. (1998). Planning and acting in partially observable stochastic domains. *Artificial Intelligence, 101*, 99–134.

Lovejoy, W. S. (1991). A survey of algorithmic methods for partially observable Markov decision processes. *Annals of Operations Research, 28*, 47–65.

McCallum, A. K. (1995). *Reinforcement learning with selective perception and hidden state.* Doctoral dissertation, Department of Computer Science, University of Rochester.

Rivest, R. L., & Schapire, R. E. (1994). Diversity-based inference of finite automata. *Journal of the ACM, 41*, 555–589.

Shatkay, H., & Kaelbling, L. P. (1997). Learning topological maps with weak local odometric information. *Proceedings of Fifteenth International Joint Conference on Artificial Intelligence (IJCAI-97)* (pp. 920–929).

Singh, S., & Cohn, D. (1998). How to dynamically merge Markov decision processes. *Advances in Neural and Information Processing Systems 10* (pp. 1057–1063).

Sutton, R. S., Precup, D., & Singh, S. (1999). Between MDPs and semi-MDPs: A framework for temporal abstraction in reinforcement learning. *Artificial Intelligence*, 181–211.

Thrun, S., & Pratt, L. (Eds.). (1998). *Learning to learn.* Kluwer Academic Publishers.

